# Speech Recognition
# Using Demi-Syllable Neural Prediction Model

Ken-ichi Iso and Takao Watanabe
C & C Information Technology Research Laboratories
NEC Corporation
4-1-1 Miyazaki, Miyamae-ku, Kawasaki 213, JAPAN

## Abstract

The Neural Prediction Model is the speech recognition model based on pattern prediction by multilayer perceptrons. Its effectiveness was confirmed by the speaker-independent digit recognition experiments. This paper presents an improvement in the model and its application to large vocabulary speech recognition, based on subword units. The improvement involves an introduction of "backward prediction," which further improves the prediction accuracy of the original model with only "forward prediction". In application of the model to speaker-dependent large vocabulary speech recognition, the demi-syllable unit is used as a subword recognition unit. Experimental results indicated a 95.2% recognition accuracy for a 5000 word test set and the effectiveness was confirmed for the proposed model improvement and the demi-syllable subword units.

## 1   INTRODUCTION

The Neural Prediction Model (NPM) is the speech recognition model based on pattern prediction by multilayer perceptrons (MLPs). Its effectiveness was confirmed by the speaker-independent digit recognition experiments (Iso, 1989; Iso, 1990; Levin, 1990).

Advantages in the NPM approach are as follows. The underlying process of the speech production can be regarded as the nonlinear dynamical system. Therefore, it is expected that there is causal relation among the adjacent speech feature vectors. In the NPM, the causality is represented by the nonlinear prediction mapping $\mathbf{F}$,

$$\mathbf{a}_t = \mathbf{F}_w(\mathbf{a}_{t-1}), \tag{1}$$

where $\mathbf{a}_t$ is the speech feature vector at frame $t$, and subscript $w$ represents mapping parameters. This causality is not explicitly considered in the conventional

speech recognition model, where the adjacent speech feature vectors are treated as independent variables.

Another important model characteristic is its applicability to continuous speech recognition. Concatenating the recognition unit models, continuous speech recognition and model training from continuous speech can be implemented without the need for segmentation.

This paper presents an improvement in the NPM and its application to large vocabulary speech recognition, based on subword units. It is an introduction of "backward prediction," which further improves the prediction accuracy for the original model with only "forward prediction". In Section 2, the improved predictor configuration, NPM recognition and training algorithms are described in detail. Section 3 presents the definition of demi-syllables used as subword recognition units. Experimental results obtained from speaker-dependent large vocabulary speech recognition are described in Section 4.

## 2  NEURAL PREDICTION MODEL

### 2.1  MODEL CONFIGURATION

Figure 1 shows the MLP predictor architecture. It is given two groups of feature vectors as input. One is feature vectors for "forward prediction". Another is feature vectors for "backward prediction". The former includes the input speech feature vectors, $\mathbf{a}_{t-\tau_F}, \ldots, \mathbf{a}_{t-1}$, which have been implemented in the original formulation. The latter, $\mathbf{a}_{t+1}, \ldots, \mathbf{a}_{t+\tau_B}$, are introduced in this paper to further improve the prediction accuracy over the original method, with only "forward prediction". This, for example, is expected to improve the prediction accuracy for voiceless stop consonants, which are characterized by a period of closure interval, followed by a sudden release. The MLP output, $\hat{\mathbf{a}}_t$, is used as a predicted feature vector for input speech

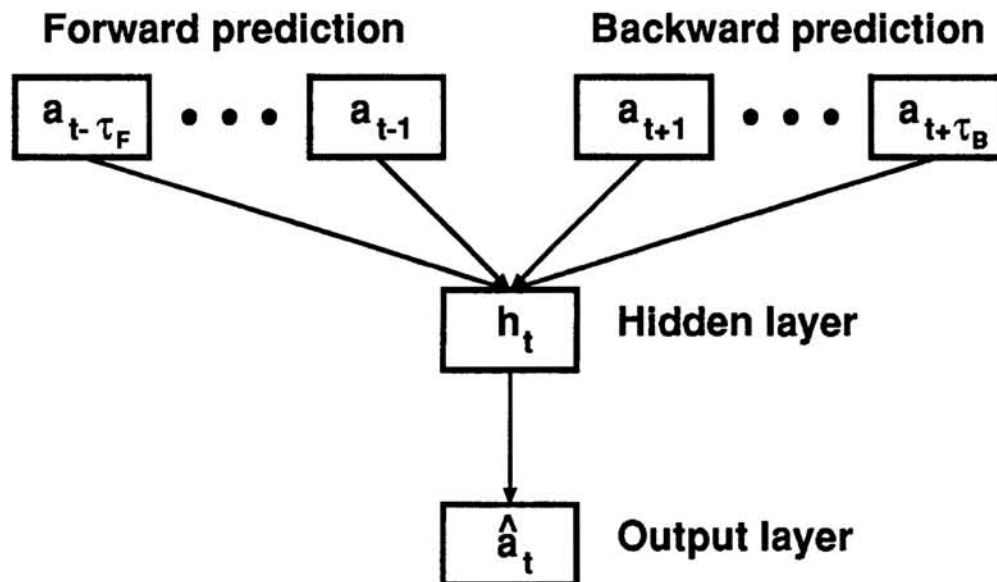

Figure 1: Multilayer perceptron predictor

feature vector $\mathbf{a}_t$. The difference between the input speech feature vector $\mathbf{a}_t$ and its prediction $\hat{\mathbf{a}}_t$ is the prediction error. Also, it can be regarded as an error function for the MLP training, based on the back-propagation technique.

The NPM for a recognition class, such as a subword unit, is constructed as a state transition network, where each state has an MLP predictor described above (Figure 2). This configuration is similar in form to the Hidden Markov Model (HMM), in which each state has a vector emission probability distribution (Rabiner, 1989). The concatenation of these subword NPMs enables continuous speech recognition.

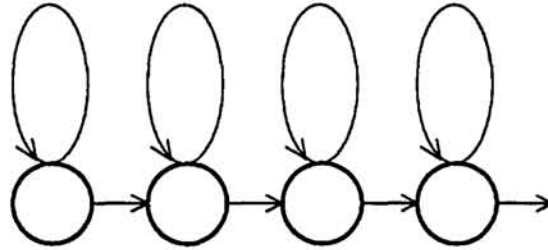

Figure 2: Neural Prediction Model

## 2.2   RECOGNITION ALGORITHM

This section presents the continuous speech recognition algorithm based on the NPM. The concatenation of subword NPMs, which is also the state transition network, is used as a reference model for the input speech. Figure 3 shows a diagram of the recognition algorithm. In the recognition, the input speech is divided into segments, whose number is equal to the total states in the concatenated NPMs ($= N$). Each state makes a prediction for the corresponding segment. The local prediction error, between the input speech at frame $t$ and the $n$-th state, is given by

$$d_t(n) = \|\mathbf{a}_t - \hat{\mathbf{a}}_t(n)\|^2, \tag{2}$$

where $n$ means the consecutive number of the state in the concatenated NPM. The accumulation of local prediction errors defines the global distance between the input speech and the concatenated NPMs

$$D = \min_{\{n_t\}} \sum_{t=1}^{T} d_t(n_t), \tag{3}$$

where $n_t$ denotes the state number used for prediction at frame $t$, and a sequence $\{n_1, n_2, \ldots, n_t, \ldots, n_T\}$ determines the segmentation of the input speech. The minimization means that the optimal segmentation, which gives a minimum accumulated prediction error, should be selected. This optimization problem can be solved by the use of dynamic-programming. As a result, the DP recursion formula is obtained

$$g_t(n) = d_t(n) + \min \left\{ \begin{array}{c} g_{t-1}(n) \\ g_{t-1}(n-1) \end{array} \right\}. \tag{4}$$

At the end of Equation (4) recursive application, it is possible to obtain $D = g_T(N)$. Backtracking the result provides the input speech segmentation.

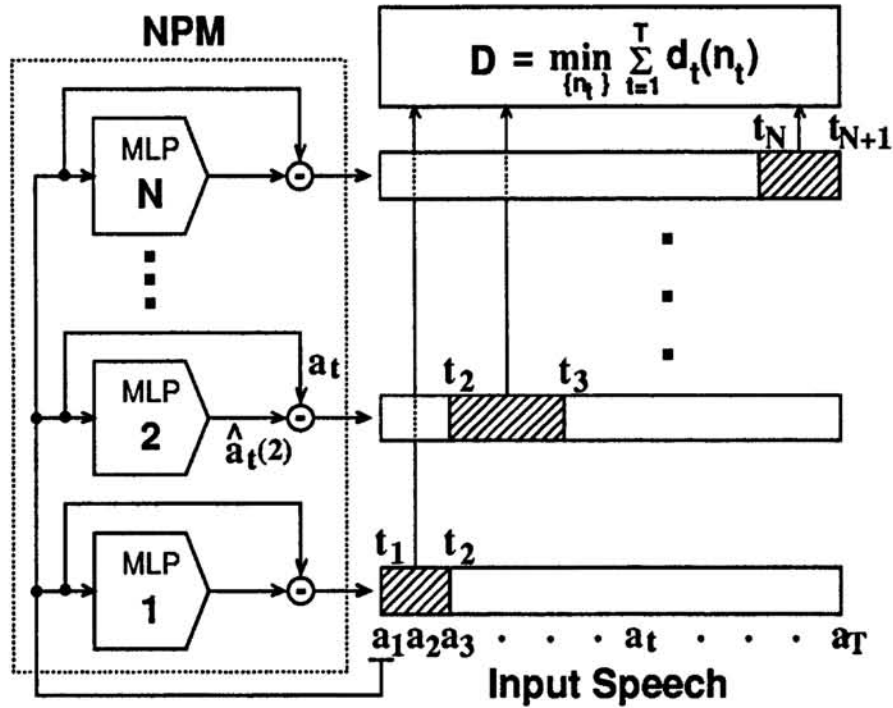

Figure 3: Recognition algorithm based on DP

In this algorithm, temporal distortion of the input speech is efficiently absorbed by DP based time-alignment between the input speech and an MLPs sequence. For simplicity, the reference model topology shown above is limited to a sequence of MLPs with no branches. It is obvious that the algorithm is applicable to more general topologies with branches.

## 2.3    TRAINING ALGORITHM

This section presents a training algorithm for estimating NPM parameters from continuous utterances. The training goal is to find a set of MLP predictor parameters, which minimizes the accumulated prediction errors for training utterances. The objective function for the minimization is defined as the average value for accumulated prediction errors for all training utterances

$$\bar{D} = \frac{1}{M} \sum_{m=1}^{M} D(m), \tag{5}$$

where $M$ is the number of training utterances and $D(m)$ is the accumulated prediction error between the $m$-th training utterance and its concatenated NPM, whose expression is given by Equation (3). The optimization can be carried out by an iterative procedure, combining dynamic-programming (DP) and back-propagation (BP) techniques. The algorithm is given as follows :

1. Initialize all MLP predictor parameters.

2. Set $m = 1$.

3. Compute the accumulated prediction error $D(m)$ by DP (Equation (4)) and determine the optimal segmentation $\{n_i^*\}$, using its backtracking.

4. Correct parameters for each MLP predictor by BP, using the optimal segmentation $\{n_i^*\}$, which determines the desired output $\mathbf{a}_t$ for the actual output $\hat{\mathbf{a}}_t(n_i^*)$ of the $n_i^*$-th MLP predictor.

5. Increase $m$ by 1.

6. Repeat 3 - 5, while $m \leq M$.

7. Repeat 2 - 6, until convergence occurs.

Convergence proof for this iterative procedure was given in (Iso, 1989; Iso, 1990). This can be intuitively understood by the fact that both DP and BP decrease the accumulated prediction error and that they are applied successively.

## 3    Demi-Syllable Recognition Units

In applying the model to large vocabulary speech recognition, the demi-syllable unit is used as a subword recognition unit (Yoshida, 1989). The demi-syllable is a half syllable unit, divided at the center of the syllable nucleus. It can treat contextual variations, caused by the co-articulation effect, with a moderate unit number. The units consist of consonant-vowel (CV) and vowel-consonant (VC) segments. Word models are made by concatenation of demi-syllable NPMs, as described in the transcription dictionary. Their segmentation boundaries are basically defined as a consonant start point and a vowel center point (Figure 4). Actually, they are automatically determined in the training algorithm, based on the minimum accumulated prediction error criterion (Section 2.3).

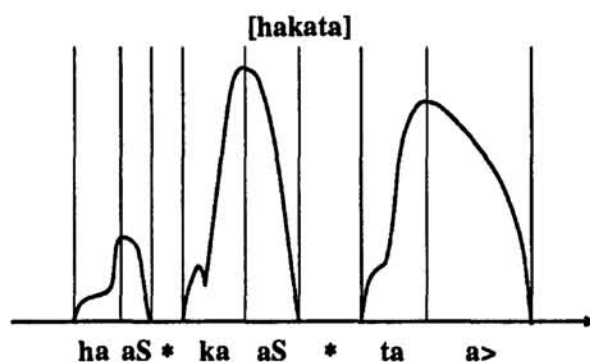

Figure 4: Demi-syllable unit boundary definition

## 4    EXPERIMENTS

### 4.1    SPEECH DATA AND MODEL CONFIGURATION

In order to examine the validity of the proposed model, speaker-dependent Japanese isolated word recognition experiments were carried out. Phonetically balanced 250, 500 and 750 word sets were selected from a Japanese word lexicon as training vocabularies. For word recognition experiments, a 250 word test set was prepared. All

the words in the test set were different from those in the training sets. A Japanese male speaker uttered these word sets in a quiet environment. The speech data was sampled at a 16 kHz sampling rate, and analyzed by a 10 msec frame period. As a feature vector for each time frame, 10 mel-scaled cepstral parameters, 10 mel-scaled delta cepstral parameters and a changing ratio parameter for amplitude were calculated from the FFT based spectrum.

The NPMs for demi-syllable units were prepared. Their total number was 241, where each demi-syllable NPM consists of a sequence of four MLP predictors, except for silence and long vowel NPMs, which have one MLP predictor. Every MLP predictor has 20 hidden units and 21 output units, corresponding to the feature vector dimensions. The numbers of input speech feature vectors, denoted by $\tau_F$, for the forward prediction, and by $\tau_B$, for the backward prediction, in Figure 1, were chosen for the two configurations, $(\tau_F, \tau_B) = (2, 1)$ and $(3, 0)$. The former, **Type A**, uses the forward and backward predictions, while the latter, **Type B**, uses the forward prediction only.

## 4.2  WORD RECOGNITION EXPERIMENTS

All possible combinations between training data amounts ($= 250, 500, 750$ words) and MLP input layer configurations (**Type A** and **Type B**) were evaluated by 5000 word recognition experiments.

To reduce the computational amount in 5000 word recognition experiments, the similar word recognition method described below was employed. For every word in the 250 word recognition vocabulary, a 100 similar word set is chosen from the 5000 word recognition vocabulary, using the distance based on the manually defined phoneme confusion matrix. In the experiments, every word in the 250 word utterances is compared with its 100 similar word set. It has been confirmed that a result approximately equivalent to actual 5000 word recognition can be obtained by this similar word recognition method (Koga, 1989).

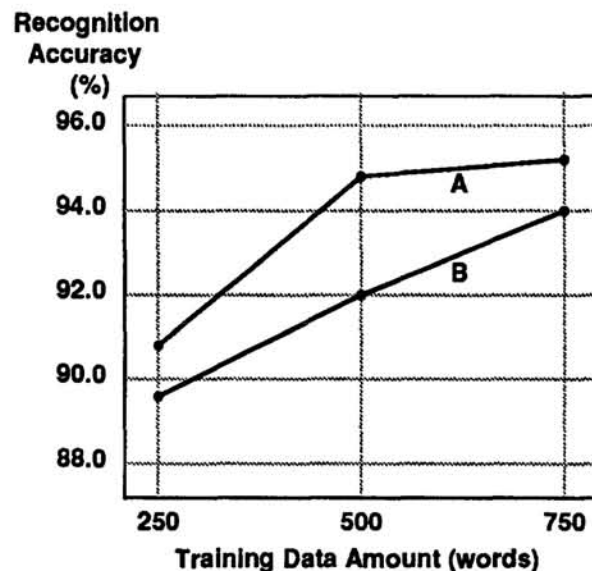

Figure 5: Recognition accuracy vs. training data amounts

The results for 5000 word recognition experiments are shown in Figure 5. As a result, consistently higher recognition accuracies were obtained for the input layer configuration with backward prediction (Type A), compared with the configuration without backward prediction (Type B), and absolute values for recognition accuracies become higher with the increase in training data amount.

## 5  DISCUSSION AND CONCLUSION

This paper presents an improvement in the Neural Prediction Model (NPM), which is the introduction of backward prediction, and its application to large vocabulary speech recognition based on the demi-syllable units. As a result of experiments, the NPM applicability to large vocabulary (5000 words) speech recognition was verified. This suggests the usefulness of the recognition and training algorithms for concatenated subword unit NPMs, without the need for segmentation. It was also reported in (Tebelskis, 1990) (90 % for 924 words), where the subword units (phonemes) were limited to a subset of complete Japanese phoneme set and the duration constraints were heuristically introduced. In this paper, the authors used the demi-syllable units, which can cover any Japanese utterances, and no duration constraints. High recognition accuracies (95.2 %), obtained for 5000 words, indicates the advantages of the use of demi-syllable units and the introduction of the backward prediction in the NPM.

## Acknowledgements

The authors wish to thank members of the Media Technology Research Laboratory for their continuous support.

## References

K. Iso. (1989), "Speech Recognition Using Neural Prediction Model," *IEICE Technical Report*, SP89-23, pp.81-87 (in *Japanese*).

K. Iso and T. Watanabe. (1990), "Speaker-Independent Word Recognition Using A Neural Prediction Model," *Proc.ICASSP-90*, S8.8, pp.441-444.

E. Levin. (1990), "Word Recognition Using Hidden Control Neural Architecture," *Proc.ICASSP-90*, S8.6, pp.433-436.

J. Tebelskis and A. Waibel. (1990), "Large Vocabulary Recognition Using Linked Predictive Neural Networks," *Proc. ICASSP-90*, S.8.7, pp.437-440.

L.R.Rabiner. (1989), "A Tutorial on Hidden Markov Models and Selected Applications in Speech Recognition", *Proc. of IEEE*, Vol.**77**, No.2, pp.257-286., February 1989.

K. Yoshida, T. Watanabe and S. Koga. (1989), "Large Vocabulary Word Recognition Based on Demi-Syllable Hidden Markov Model Using Small Amount of Training Data," *Proc.ICASSP-89*, S1.1, pp.1-4.

S. Koga, K. Yoshida, and T. Watanabe. (1989), "Evaluation of Large Vocabulary Speech Recognition Based on Demi-Syllable HMM," *Proc. of ASJ Autumn Meeting* (in Japanese).